# EEG-Based Brain-Computer Interaction: Improved Accuracy by Automatic Single-Trial Error Detection

**Pierre W. Ferrez**
IDIAP Research Institute
Centre du Parc
Av. des Prés-Beudin 20
1920 Martigny, Switzerland
pierre.ferrez@idiap.ch

José del R. Millán
IDIAP Research Institute
Centre du Parc
Av. des Prés-Beudin 20
1920 Martigny, Switzerland
jose.millan@idiap.ch *

## Abstract

Brain-computer interfaces (BCIs), as any other interaction modality based on physiological signals and body channels (e.g., muscular activity, speech and gestures), are prone to errors in the recognition of subject's intent. An elegant approach to improve the accuracy of BCIs consists in a verification procedure directly based on the presence of error-related potentials (ErrP) in the EEG recorded right after the occurrence of an error. Six healthy volunteer subjects with no prior BCI experience participated in a new human-robot interaction experiment where they were asked to mentally move a cursor towards a target that can be reached within a few steps using motor imagination. This experiment confirms the previously reported presence of a new kind of ErrP. These "Interaction ErrP" exhibit a first sharp negative peak followed by a positive peak and a second broader negative peak ($\sim$290, $\sim$350 and $\sim$470 ms after the feedback, respectively). But in order to exploit these ErrP we need to detect them in each single trial using a short window following the feedback associated to the response of the classifier embedded in the BCI. We have achieved an average recognition rate of correct and erroneous single trials of 81.8% and 76.2%, respectively. Furthermore, we have achieved an average recognition rate of the subject's intent while trying to mentally drive the cursor of 73.1%. These results show that it's possible to simultaneously extract useful information for mental control to operate a brain-actuated device as well as cognitive states such as error potentials to improve the quality of the brain-computer interaction. Finally, using a well-known inverse model (sLORETA), we show that the main focus of activity at the occurrence of the ErrP are, as expected, in the pre-supplementary motor area and in the anterior cingulate cortex.

## 1 Introduction

People with severe motor disabilities (spinal cord injury (SCI), amyotrophic lateral sclerosis (ALS), etc.) need alternative ways of communication and control for their everyday life. Over the past two decades, numerous studies proposed electroencephalogram (EEG) activity for direct brain-computer interaction [1]-[2]. EEG-based brain-computer interfaces (BCIs) provide disabled people with new tools for control and communication and are promising alternatives to invasive methods. However, as any other interaction modality based on physiological signals and body channels (e.g., muscular activity, speech and gestures), BCIs are prone to errors in the recognition of subject's intent, and those errors can be frequent. Indeed, even well-trained subjects rarely reach 100% of success. In

contrast to other interaction modalities, a unique feature of the "brain channel" is that it conveys both information from which we can derive mental control commands to operate a brain-actuated device as well as information about cognitive states that are crucial for a purposeful interaction, all this on the millisecond range. One of these states is the awareness of erroneous responses, which a number of groups have recently started to explore as a way to improve the performance of BCIs [3]-[6].

In particular, [6] recently reported the presence of a new kind of error potentials (ErrP) elicited by erroneous feedback provided by a BCI during the recognition of the subject's intent. In this study subjects were asked to reach a target by sending repetitive manual commands to pass over several steps. The system was executing commands with an 80% accuracy, so that at each step there was a 20% probability that the system delivered an erroneous feedback. The main components of these "Interaction ErrP" are a negative peak 250 ms after the feedback, a positive peak 320 ms after the feedback and a second broader negative peak 450 ms after the feedback. To exploit these ErrP for BCIs, it is mandatory to detect them no more in grand averages but in each single trial using a short window following the feedback associated to the response of the BCI. The reported average recognition rates of correct and erroneous single trials are 83.5% and 79.2%, respectively. These results tend to show that ErrP could be a potential tool to improve the quality of the brain-computer interaction. However, it is to note that in order to isolate the issue of the recognition of ErrP out of the more difficult and general problem of a whole BCI where erroneous feedback can be due to non-optimal performance of both the interface (i.e., the classifier embedded into the interface) and the user himself, the subject delivered commands manually. The key issue now is to investigate whether subjects also show ErrP while already engaged in tasks that require a high level of concentration such as motor imagination, and no more in easy tasks such as pressing a key.

The objective of the present study is to investigate the presence of these ErrP in a real BCI task. Subjects don't deliver manual commands anymore, but are focussing on motor imagination tasks to reach targets randomly selected by the system. In this paper we report new experimental results recorded with six healthy volunteer subjects with no prior BCI experience during a simple human-robot interaction that confirm the previously reported existence of a new kind of ErrP [6], which is satisfactorily recognized in single trials using a short window just after the feedback. Furthermore, using a window just before the feedback, we report a 73.1% accuracy in the recognition of the subject's intent during mental control of the BCI. This confirms the fact that EEG conveys simultaneously information from which we can derive mental commands as well as information about cognitive states and shows that both can be sufficiently well recognized in each single trials to provide the subject with an improved brain-computer interaction. Finally, using a well-known inverse model called sLORETA [7] that non-invasively estimates the intracranial activity from scalp EEG, we show that the main focus of activity at the occurrence of ErrP seems to be located in the pre-supplementary motor area (pre-SMA) and in the anterior cingulate cortex (ACC), as expected [8][9].

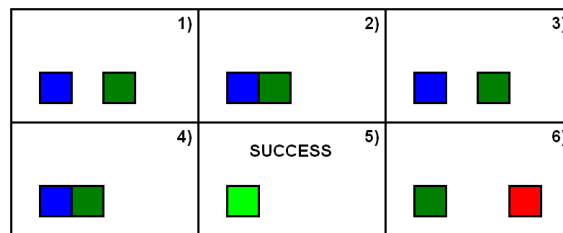

Figure 1: Illustration of the protocol. (1) The target (blue) appears 2 steps on the left side of the cursor (green). (2) The subject is imagining a movement of his/her left hand and the cursor moves 1 step to the left. (3) The subject still focuses on his/her left hand, but the system moves the cursor in the wrong direction. (4) Correct move to the left, compensating the error. (5) The cursor reaches the target. (6) A new target (red) appears 3 steps on the right side of the cursor, the subject will now imagine a movement of his/her right foot. The system moved the cursor with an error rate of 20%; i.e., at each step, there was a 20% probability that the robot made a movement in the wrong direction.

## 2   Experimental setup

The first step to integrate ErrP detection in a BCI is to design a protocol where the subject is focussing on a mental task for device control and on the feedback delivered by the BCI for ErrP

detection. To test the ability of BCI users to concentrate simultaneously on a mental task and to be aware of the BCI feedback at each single trial, we have simulated a human-robot interaction task where the subject has to bring the robot to targets 2 or 3 steps either to the left or to the right. This virtual interaction is implemented by means of a green square cursor that can appear on any of 20 positions along an horizontal line. The goal with this protocol is to bring the cursor to a target that randomly appears either on the left (blue square) or on the right(red square) of the cursor. The target is no further away than 3 positions from the cursor (symbolizing the current position of the robot). This prevents the subject from habituation to one of the stimuli since the cursor reaches the target within a small number of steps. Figure 1 illustrates the protocol with the target (blue) initially positioned 2 steps away on the left side of the cursor (green). An error occurred at step 3) so that the cursor reaches the target in 5 steps. Each target corresponds to a specific mental task. The subjects were asked to imagine a movement of their left hand for the left target and to imagine a movement of their right foot for the right target (note that subject n°1 selected left foot for the left target and right hand for the right target). However, since the subjects had no prior BCI experience, the system was not moving the cursor following the mental commands of the subject, but with an error rate of 20%, to avoid random or totally biased behavior of the cursor.

Six healthy volunteer subjects with no prior BCI experience participated in these experiments. After the presentation of the target, the subject focuses on the corresponding mental task until the cursor reached the target. The system moved the cursor with an error rate of 20%; i.e., at each step, there was a 20% probability that the cursor moved in the opposite direction. When the cursor reached a target, it briefly turned from green to light green and then a new target was randomly selected by the system. If the cursor didn't reach the target after 10 steps, a new target was selected. As shown in figure 2, while the subject focuses on a specific mental task, the system delivers a feedback about every 2 seconds. This provides a window just before the feedback for BCI classification and a window just after the feedback for ErrP detection for every single trial. Subjects performed 10 sessions of 3 minutes on 2 different days (the delay between the two days of measurements varied from 1 week to 1 month), corresponding to ~75 single trials per session. The 20 sessions were split into 4 groups of 5, so that classifiers were built using a group and tested on the following group. The classification rates presented in Section 3 are therefore the average of 3 prediction performances: classification of group $n + 1$ using group $n$ to build a classifier. This rule applies for both mental tasks classification and ErrP detection.

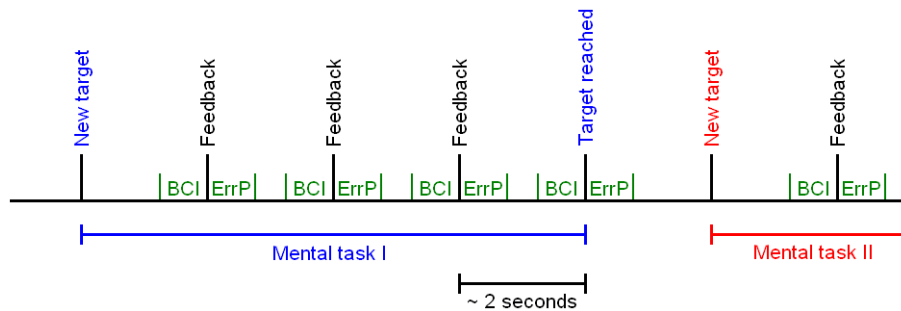

Figure 2: Timing of the protocol. The system delivers a feedback about every 2 seconds, this provides a window just before the feedback for BCI classification and a window just after the feedback for ErrP detection for every single trial. As a new target is presented, the subject focuses on the corresponding mental task until the target is reached.

EEG potentials were acquired with a portable system (Biosemi ActiveTwo) by means of a cap with 64 integrated electrodes covering the whole scalp uniformly. The sampling rate was 512 Hz and signals were measured at full DC. Raw EEG potentials were first spatially filtered by subtracting from each electrode the average potential (over the 64 channels) at each time step. The aim of this re-referencing procedure is to suppress the average brain activity, which can be seen as underlying background activity, so as to keep the information coming from local sources below each electrode. Then for off-line mental tasks classification, the power spectrum density (PSD) of EEG channels was estimated over a window of one second just before the feedback. PSD was estimated using the Welch method resulting in spectra with a 2 Hz resolution from 6 to 44 Hz. The most relevant EEG channels and frequencies were selected by a simple feature selection algorithm based on the overlap of the distributions of the different classes. For off-line ErrP detection, we applied a 1-10

Hz bandpass filter as ErrP are known to be a relatively slow cortical potential. EEG signals were then subsampled from 512 Hz to 64 Hz (i.e., we took one point out of 8) before classification, which was entirely based on temporal features. Indeed the actual input vector for the statistical classifier described below is a 150 ms window starting 250 ms after the feedback for channels FCz and Cz. The choice of these channels follows the fact that ErrP are characterized by a fronto-central distribution along the midline.

For both mental tasks and ErrP classification, the two different classes (left or right for mental tasks and error or correct for ErrP) are recognized by a Gaussian classifier. The output of the statistical classifier is an estimation of the posterior class probability distribution for a single trial; i.e., the probability that a given single trial belongs to one of the two classes. In this statistical classifier, every Gaussian unit represents a prototype of one of the classes to be recognized, and we use several prototypes per class. During learning, the centers of the classes of the Gaussian units are pulled towards the trials of the class they represent and pushed away from the trials of the other class. No artifact rejection algorithm (for removing or filtering out eye or muscular movements) was applied and all trials were kept for analysis. It is worth noting, however, that after a visual a-posteriori check of the trials we found no evidence of muscular artifacts that could have contaminated one condition differently from the other. More details on the Gaussian classifier and the analysis procedure to rule out ocular/muscular artifacts as the relevant signals for both classifiers (BCI itself and ErrP) can be found in [10].

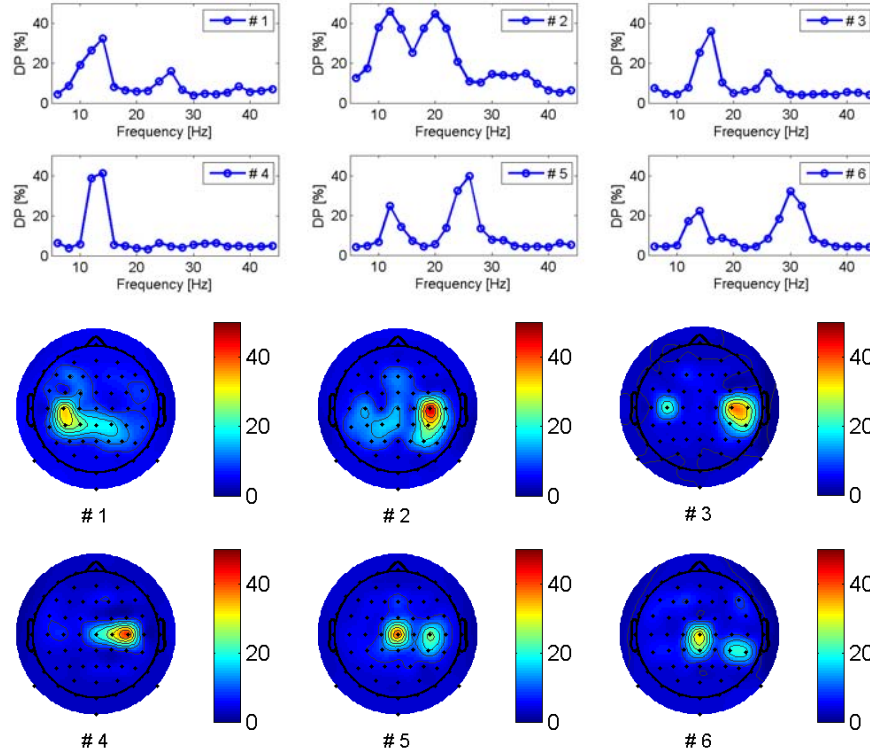

Figure 3: (*Top*) Discriminant power (DP) of frequencies. Sensory motor rhythm (12-16 Hz) and some beta components are discriminant for all subjects. (*Bottom*) Discriminant power (DP) of electrodes. The most relevant electrodes are in the central area (C3, C4 and Cz) according to the ERD/ERD location for hand and foot movement or imagination.

## 3 Experimental results

### 3.1 Mental tasks classification

Subject were asked to imagine a movement of their left hand when the left target was proposed and to imagine a movement of their right foot when the right target was proposed (note that subject n°1

was imagining left foot for the left target and right hand for the right target). The most relevant EEG channels and frequencies were selected by a simple feature selection algorithm based on the overlap of the distributions of the different classes. Figure 3 shows the discriminant power (DP) of frequencies (top) and electrodes (bottom) for the 6 subject. For frequencies, the DP is based on the best electrode, and for electrodes it is based on the best frequency. Table 1 shows the classification rates for the two mental tasks and the general BCI accuracy for the 6 subjects and the average of them, it also shows the features (electrodes and frequencies) used for classification.

For all 6 subjects, the 12-16 Hz band (sensory motor rhythm (SMR)) appears to be relevant for classification. Subject 1, 3 and 5 show a peak in DP for frequencies around 25 Hz (beta band). For subject 2 this peak in the beta band is centered at 20 Hz and for subject 6 it is centered at 30 Hz. Finally subject 4 shows no particular discriminant power in the beta band. Previous studies confirm these results. Indeed, SMR and beta rhythm over left and/or right sensorimotor cortex have been successfully used for BCI control [11]. Event-related de-synchronization (ERD) and synchronization (ERS) refer to large-scale changes in neural processing. During periods of inactivity, brain areas are in a kind of idling state with large populations of neurons firing in synchrony resulting in an increase of amplitude of specific alpha (8-12 Hz) and beta (12-26 Hz) bands. During activity, populations of neurons work at their own pace and the power of this idling state is reduced, the cortex has become de-synchronized. [12]. In our case, the most relevant electrodes for all subjects are in the C3, C4 or Cz area. These locations confirm previous studies since C3 and C4 areas usually show ERD/ERS during hands movement or imagination whereas foot movement or imagination are focused in the Cz area [12].

Table 1: Percentages (mean and standard deviations) of correctly recognized single trials for the 2 motor imagination tasks for the 6 subjects and the average of them. All subjects show classification rates of about 70-75% for motor imagination and the general BCI accuracy is 73%. Features used for classification are also shown.

| | Electrodes | Frequencies [Hz] | Left hand [%] | Right foot [%] | Accuracy [%] |
|---|---|---|---|---|---|
| # 1* | C3 CP3 CP1 CPz CP2 | 10 12 14 26 | $77.2 \pm 3.7$ | $70.4 \pm 3.2$ | $\mathbf{73.8 \pm 4.8}$ |
| # 2 | C4 CP4 P4 | 10 12 14 18 20 22 | $71.8 \pm 9.0$ | $80.9 \pm 7.1$ | $\mathbf{76.4 \pm 6.4}$ |
| # 3 | C3 C4 C6 CP6 CP4 | 14 16 26 | $76.4 \pm 5.8$ | $62.6 \pm 6.7$ | $\mathbf{69.5 \pm 9.8}$ |
| # 4 | Cz C2 C4 | 12 14 | $79.6 \pm 1.6$ | $66.3 \pm 10.1$ | $\mathbf{73.0 \pm 9.4}$ |
| # 5 | Cz C4 CP4 | 12 24 26 | $73.5 \pm 16.1$ | $71.9 \pm 13.3$ | $\mathbf{72.7 \pm 1.1}$ |
| # 6 | CPz Cz CP6 CP4 | 12 14 28 30 32 | $77.9 \pm 7.4$ | $69.0 \pm 13.7$ | $\mathbf{73.5 \pm 6.3}$ |
| Avg | | | $\mathbf{76.1 \pm 2.9}$ | $\mathbf{70.2 \pm 6.2}$ | $\mathbf{73.1 \pm 4.2}$ |

* Left foot and Right hand

All 6 subjects show classification rates of about 70-75% for motor imagination. These figures were achieved with a relatively low number of features (up to 5 electrodes and up to 6 frequencies) and the general BCI accuracy is 73%. This level of performance can appear relatively low for a 2-class BCI. However, keeping in mind that first all subjects had no prior BCI experience and second that these figures were obtained exclusively in prediction (i.e. classifiers were always tested on new data), the performance is satisfactory.

## 3.2 Error-related potentials

Figure 4 shows the averages of error trials (red curve), of correct trials (green curve) and the difference error-minus-correct (blue curve) for channel FCz for the six subjects (top). A first small positive peak shows up about ∼230 ms after the feedback (t=0). A negative peak clearly appears ∼290 ms after the feedback for 5 subjects. This negative peak is followed by a positive peak ∼350 ms after the feedback. Finally a second broader negative peak occurs about ∼470 ms after the feedback. Figure 4 also shows the scalp potentials topographies (right) for the average of the six subjects, at the occurrence of the four previously described peaks: a first fronto-central positivity appears after ∼230 ms, followed by a fronto-central negativity at ∼290 ms, a fronto-central positivity at ∼350 ms and a fronto-central negativity at ∼470 ms. All six subjects show similar ErrP time courses whose amplitudes slightly differ from one subject to the other. These experiments seem to confirm the existence of a new kind of error-related potentials [6]. Furthermore, the fronto-central

focus at the occurrence of the different peaks tends to confirm the hypothesis that ErrP are generated in a deep brain region called anterior cingulate cortex [8][9] (see also Section 3.3).

Table 2 reports the recognition rates (mean and standard deviations) for the six subjects plus the average of them. These results show that single-trial recognition of erroneous and correct responses are above 75% and 80%, respectively. Beside the crucial importance to integrate ErrP in the BCI in a way that the subject still feels comfortable, for example by reducing as much as possible the rejection of actually correct commands, a key point for the exploitation of the automatic recognition of interaction errors is that they translate into an actual improvement of the performance of the BCI. Table 2 also show the performance of the BCI in terms of bit rate (bits per trial) when detection of ErrP is used or not and the induced increase of performance (for details see [6]). The benefit of integrating ErrP detection is obvious since it at least doubles the bit rate for five of the six subjects and the average increase is 124%.

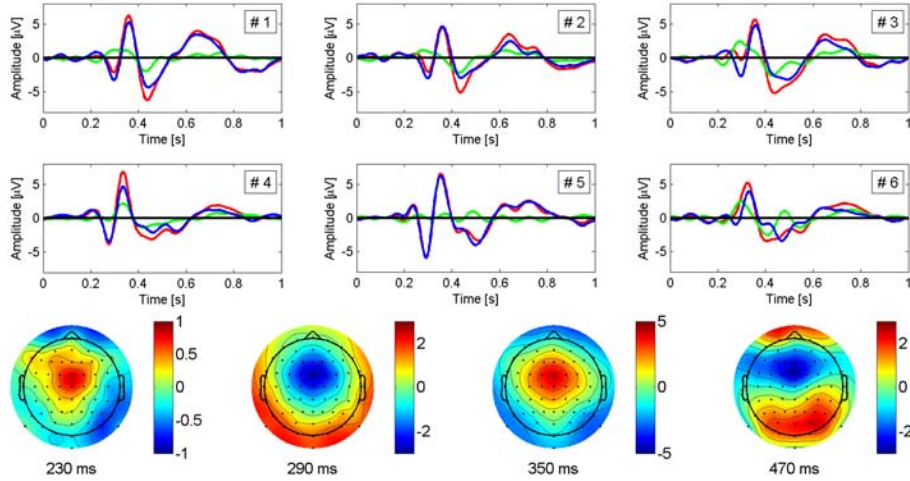

Figure 4: (*Top*) Averages of error trials (red curve), of correct trials (green curve) and the difference error-minus-correct (blue curve) for channel FCz for the six subjects. All six subjects show similar ErrP time courses whose amplitudes slightly differ from one subject to the other. (*Bottom*) Scalp potentials topographies for the average of the six subjects, at the occurrence of the four described peaks. All focuses are located in fronto-central areas, over the anterior cingulate cortex (ACC).

Table 2: Percentages (mean and standard deviations) of correctly recognized error trials and correct trials for the six subjects and the average of them. Table also show the BCI performance in terms of bit rate and its increase using ErrP detection. Classification rates are above 75% and 80% for error trials and correct trials, respectively. The benefit of integrating ErrP detection is obvious since it at least doubles the bit rate for five of the six subjects.

|  | **Error** [%] | **Correct** [%] | **BCI accuracy** [%] (from Table 1) | **Bit rate** [bits/trial] (no ErrP) | (ErrP) | **Increase** [%] |
|---|---|---|---|---|---|---|
| **# 1** | $77.7 \pm 13.9$ | $76.8 \pm 5.4$ | $73.8 \pm 4.8$ | 0.170 | 0.345 | **103** |
| **# 2** | $75.4 \pm 5.5$ | $80.1 \pm 7.9$ | $76.4 \pm 6.4$ | 0.212 | 0.385 | **82** |
| **# 3** | $74.0 \pm 12.9$ | $85.9 \pm 1.6$ | $69.5 \pm 9.8$ | 0.113 | 0.324 | **187** |
| **# 4** | $84.3 \pm 7.7$ | $80.1 \pm 5.5$ | $73.0 \pm 9.4$ | 0.159 | 0.403 | **154** |
| **# 5** | $75.3 \pm 6.0$ | $85.6 \pm 5.2$ | $72.7 \pm 1.1$ | 0.154 | 0.371 | **141** |
| **# 6** | $70.7 \pm 11.4$ | $82.2 \pm 5.1$ | $73.5 \pm 6.3$ | 0.166 | 0.333 | **101** |
| **Avg** | **$76.2 \pm 4.6$** | **$81.8 \pm 3.5$** | **$73.1 \pm 4.2$** | **0.160** | **0.359** | **124** |

## 3.3 Estimation of intracranial activity

Estimating the neuronal sources that generate a given potential map at the scalp surface (EEG) requires the solution of the so-called inverse problem. This inverse problem is always initially undetermined, i.e. there is no unique solution since a given potential map at the surface can be

generated by many different intracranial activity map. The inverse problem requires supplementary a priori constraints in order to be univocally solved. The ultimate goal is to unmix the signals measured at the scalp and to attribute to each brain area its own estimated temporal activity. The sLORETA inverse model [7] is a standardized low resolution brain electromagnetic tomography. This software, known for its zero localization error, was used as a localization tool to estimate the focus of intracranial activity at the occurrence of the four ErrP peaks described in Section 3.2. Figure 5 shows Talairach slices of localized activity for the grand average of the six subjects at the occurrence of the four described peaks and at the occurrence of a late positive component showing up 650 ms after the feedback. As expected, the areas involved in error processing, namely the pre-supplementary motor area (pre-SMA, Brodmann area 6) and the rostral cingulate zone (RCZ, Brodmann areas 24 & 32) are systematically activated [8][9]. For the second positive peak (350 ms) and mainly for the late positive component (650 ms), parietal areas are also activated. These associative areas (somatosensory association cortex, Brodmann areas 5 & 7) could be related to the fact that the subject becomes aware of the error. It has been proposed that the positive peak was associated with conscious error recognition in case of error potentials elicited in reaction task paradigm [13]. In our case, activation of parietal areas after 350 ms after the feedback agrees with this hypothesis.

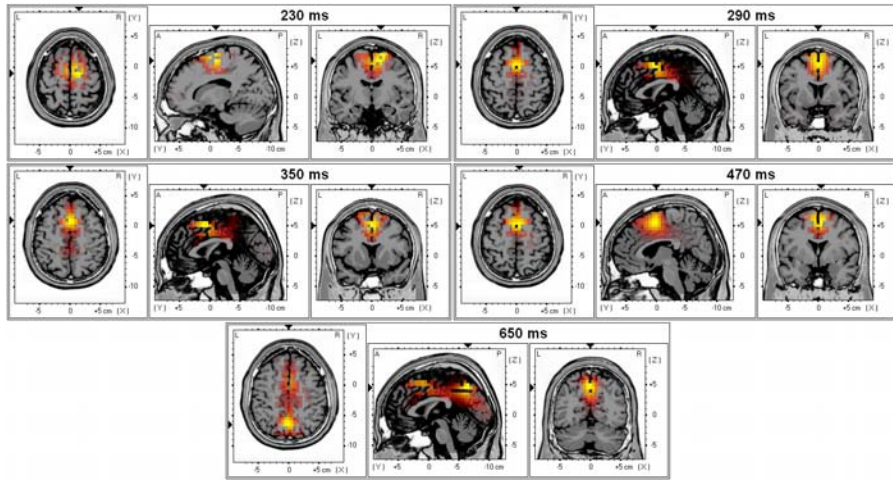

Figure 5: Talairach slices of localized activity for the grand average of the six subjects at the occurrence of the four peaks described in Section 3.2 and at the occurrence of a late positive component showing up 650 ms after the feedback. Supplementary motor cortex and anterior cingulate cortex are systematically activated. Furthermore, for the second positive peak (350 ms) and mainly for the late positive component (650 ms), parietal areas are also activated. This parietal activation could reflect the fact that the subject is aware of the error.

## 4 Discussion

In this study we have reported results on the detection of the neural correlate of error awareness for improving the performance and reliability of BCI. In particular, we have confirmed the existence of a new kind of error-related potential elicited in reaction to an erroneous recognition of the subject's intention. More importantly, we have shown the feasibility of simultaneously and satisfactorily detecting erroneous responses of the interface and classifying motor imagination for device control at the level of single trials. However, the introduction of an automatic response rejection strongly interferes with the BCI. The user needs to process additional information which induces higher workload and may considerably slow down the interaction. These issues have to be investigated when running online BCI experiments integrating automatic error detection. Given the promising results obtained in this simulated human-robot interaction, we are currently working in the actual integration of online ErrP detection into our BCI system. The preliminary results are very promising and confirm that the online detection of errors is a tool of great benefit, especially for subjects with no prior BCI experience or showing low BCI performance. In parallel, we are exploring how to increase the recognition rate of single-trial erroneous and correct responses.

In this study we have also shown that, as expected, typical cortical areas involved in error processing such as pre-supplementary motor area and anterior cingulate cortex are systematically activated at the occurrence of the different peaks. The software used for the estimation of the intracranial activity (sLORETA) is only a localization tool. However, Babiloni et al. [14] have recently developed the so-called CCD ("cortical current density") inverse model that estimates the activity of the cortical mantle. Since ErrP seems to be generated by cortical areas, we plan to use this method to best discriminate erroneous and correct responses of the interface. As a matter of fact, a key issue to improve classification is the selection of the most relevant current dipoles out of a few thousands. In fact, the very preliminary results using the CCD inverse model confirm the reported localization in the pre-supplementary motor area and in the anterior cingulate cortex and thus we may well expect a significant improvement in recognition rates by focusing on the dipoles estimated in those specific brain areas.

More generally, the work described here suggests that it could be possible to recognize in real time high-level cognitive and emotional states from EEG (as opposed, and in addition, to motor commands) such as alarm, fatigue, frustration, confusion, or attention that are crucial for an effective and purposeful interaction. Indeed, the rapid recognition of these states will lead to truly adaptive interfaces that customize dynamically in response to changes of the cognitive and emotional/affective states of the user.

## Footnotes

*This work is supported by the European IST Programme FET Project FP6-003758 and by the Swiss National Science Foundation NCCR "IM2". This paper only reflects the authors' views and funding agencies are not liable for any use that may be made of the information contained herein.

# References

[1] J.R. Wolpaw, N. Birbaumer, D.J. McFarland, G. Pfurtscheller, and T.M. Vaughan. Brain-computer interfaces for communication and control. *Clinical Neurophysiology*, 113:767–791, 2002.

[2] J. del R. Millán, F. Renkens, J. Mouriño, and W. Gerstner. Non-invasive brain-actuated control of a mobile robot by human EEG. *IEEE Transactions on Biomedical Engineering*, 51:1026–1033, 2004.

[3] G. Schalk, J.R. Wolpaw, D.J. McFarland, and G. Pfurtscheller. EEG-based communication: presence of and error potential. *Clinical Neurophysiology*, 111:2138–2144, 2000.

[4] B. Blankertz, G. Dornhege, C. Schäfer, R. Krepki, J. Kohlmorgen, K.-R. Müller, V. Kunzmann, F. Losch, and G. Curio. Boosting bit rates and error detection for the classification of fast-paced motor commands based on single-trial EEG analysis. *IEEE Transactions on Neural Systems and Rehabilitation Engineering*, 11(2):127–131, 2003.

[5] L.C. Parra, C.D. Spence, A.D. Gerson, and P. Sajda. Response error correction—a demonstration of improved human-machine performance using real-time EEG monitoring. *IEEE Transactions on Neural Systems and Rehabilitation Engineering*, 11(2):173–177, 2003.

[6] P.W. Ferrez and J. del R. Millán. You are wrong!—Automatic detection of interaction errors from brain waves. In *Proc. 19th Int. Joint Conf. Artificial Intelligence*, 2005.

[7] R.D. Pascual-Marqui. Standardized low resolution brain electromagnetic tomography (sLORETA): Technical details. *Methods & Findings in Experimental & Clinical Pharmacology*, 24D:5–12, 2002.

[8] C.B. Holroyd and M.G.H. Coles. The neural basis of human error processing: Reinforcement learning, dopamine and the error-related negativity. *Psychological Review*, 109:679–709, 2002.

[9] K. Fiehler, M. Ullsperger, and Y. von Cramon. Neural correlates of error detection and error correction: Is there a common neuroanatomical substrate? *European Journal of Neuroscience*, 19:3081–3087, 2004.

[10] P.W. Ferrez and J. del R. Millán. Error-related EEG potentials in brain-computer interfaces. In G. Dornhege, J. del R. Millán, T. Hinterberger, D. McFarland, and K.-R. Müller, editors, *Toward Brain-Computing Interfacing*, pages 291–301. The MIT Press, 2007.

[11] D. McFarland and J.R. Wolpow. Sensorimotor rhythm-based brain-computer interface (BCI): Feature selection by regression improves performance. *IEEE Transactions on Neural Systems and Rehabilitation Engineering*, 13(3):372–379, 2005.

[12] G. Pfurtscheller and F.H. Lopes da Silva. Event-related EEG/MEG synchronization and desynchronization: Basic principles. *Clinical Neurophysiology*, 110:1842–1857, 1999.

[13] S. Nieuwenhuis, K.R. Ridderinkhof, J. Blom, G.P.H. Band, and A. Kok. Error-related brain potentials are differently related to awareness of response errors: Evidence from an antisaccade task. *Psychophysiology*, 38:752–760, 2001.

[14] F. Babiloni, C. Babiloni, L. Locche, F. Cincotti, P.M. Rossini, and F. Carducci. High-resolution electroencephalogram: Source estimates of laplacian-transformed somatosensory-evoked potentials using realistic subject head model constructed from magnetic resonance imaging. *Medical & Biological Engineering and Computing*, 38:512–519, 2000.

